# Scaled Gradients on Grassmann Manifolds for Matrix Completion

**Thanh T. Ngo and Yousef Saad**
Department of Computer Science and Engineering
University of Minnesota, Twin Cities
Minneapolis, MN 55455
thango@cs.umn.edu, saad@cs.umn.edu

## Abstract

This paper describes gradient methods based on a scaled metric on the Grassmann manifold for low-rank matrix completion. The proposed methods significantly improve canonical gradient methods, especially on ill-conditioned matrices, while maintaining established global convegence and exact recovery guarantees. A connection between a form of subspace iteration for matrix completion and the scaled gradient descent procedure is also established. The proposed conjugate gradient method based on the scaled gradient outperforms several existing algorithms for matrix completion and is competitive with recently proposed methods.

## 1 Introduction

Let $A \in \mathbb{R}^{m \times n}$ be a rank-$r$ matrix, where $r \ll m, n$. The matrix completion problem is to reconstruct $A$ given a subset of entries of $A$. This problem has attracted much attention recently [8, 14, 13, 18, 21] because of its broad applications, e.g., in recommender systems, structure from motion, and multitask learning (see e.g. [19, 9, 2]).

### 1.1 Related work

Let $\Omega = \{(i,j)|A_{ij}$ is observed$\}$. We define $P_\Omega(A) \in \mathbb{R}^{m \times n}$ to be the projection of $A$ onto the observed entries $\Omega$: $P_\Omega(A)_{ij} = A_{ij}$ if $(i,j) \in \Omega$ and $P_\Omega(A)_{ij} = 0$ otherwise. If the rank is unknown and there is no noise, the problem can be formulated as:

$$\text{Minimize rank}(X) \text{ subject to } P_\Omega(X) = P_\Omega(A). \tag{1}$$

Rank minimization is NP-hard and so work has been done to solve a convex relaxation of it by approximating the rank by the nuclear norm. Under some conditions, the solution of the relaxed problem can be shown to be the exact solution of the rank minimization problem with overwhelming probability [8, 18]. Usually, algorithms to minimize the nuclear norm iteratively use the Singular Value Decomposition (SVD), specifically the singular value thresholding operator [7, 15, 17], which makes them expensive.

If the rank is known, we can formulate the matrix completion problem as follows:

$$\text{Find matrix } X \text{ to minimize } ||P_\Omega(X) - P_\Omega(A)||_F \text{ subject to rank}(X) = r. \tag{2}$$

Keshavan et al. [14] have proved that exact recovery can be obtained with high probability by solving a non-convex optimization problem. A number of algorithms based on non-convex formulation use the framework of optimization on matrix manifolds [14, 22, 6]. Keshavan et al. [14] propose a steepest descent procedure on the product of Grassmann manifolds of $r$-dimensional subspaces. Vandereycken [22] discusses a conjugate gradient algorithm on the Riemann manifold of rank-$r$ matrices. Boumal and Absil [6] consider a trust region method on the Grassmann manifold. Although

they do not solve an optimization problem on the matrix manifold, Wei et al. [23] perform a low rank matrix factorization based on a successive over-relaxation iteration. Also, Srebro and Jaakkola [21] discuss SVD-EM, one of the early fixed-rank methods using truncated singular value decomposition iteratively. Dai et al. [10] recently propose an interesting approach that does not use the Frobenius norm of the residual as the objective function but instead uses the consistency between the current estimate of the column space (or row space) and the observed entries. Guaranteed performance for this method has been established for rank-1 matrices.

In this paper, we will focus on the case when the rank $r$ is known and solve problem (2). In fact, even when the rank is unknown, the sparse matrix which consists of observed entries can give us a very good approximation of the rank based on its singular spectrum [14]. Also, a few values of the rank can be used and the best one is selected. Moreover, the singular spectrum is revealed during the iterations, so many fixed rank methods can also be adapted to find the rank of the matrix.

### 1.2 Our contribution

OptSpace [14] is an efficient algorithm for low-rank matrix completion with global convergence and exact recovery guarantees. We propose using a non-canonical metric on the Grassmann manifold to improve OptSpace while maintaining its appealing properties. The non-canonical metric introduces a scaling factor to the gradient of the objective function which can be interpreted as an adaptive preconditioner for the matrix completion problem. The gradient descent procedure using the scaled gradient is related to a form of subspace iteration for matrix completion. Each iteration of the subspace iteration is inexpensive and the procedure converges very rapidly. The connection between the two methods leads to some improvements and to efficient implementations for both of them.

Throughout the paper, $A_\Omega$ will be a shorthand for $P_\Omega(A)$ and qf$(U)$ is the $Q$ factor in the QR factorization of $U$ which gives an orthonormal basis for span $(U)$. Also, $P_{\bar{\Omega}}(.)$ denotes the projection onto the negation of $\Omega$.

## 2 Subspace iteration for incomplete matrices

We begin with a form of subspace iteration for matrix completion depicted in Algorithm 1. If the

---

**Algorithm 1** SUBSPACE ITERATION FOR INCOMPLETE MATRICES.

---

**Input:** Matrix $A_\Omega$, $\Omega$, and the rank $r$.
**Output:** Left and right dominant subspaces $U$ and $V$ and associated singular values.
1: $[U_0, \Sigma_0, V_0] = \text{svd}(A_\Omega, r)$, $S_0 = \Sigma_0$;          // *Initialize U, V and $\Sigma$*
2: **for** $i = 0,1,2,...$ **do**
3:          $X_{i+1} = P_{\bar{\Omega}}(U_i S_i V_i^T) + A_\Omega$          // *Obtain new estimate of A*
4:          $U_{i+1} = X_{i+1} V_i$;   $V_{i+1} = X_{i+1}^T U_{i+1}$          // *Update subspaces*
5:          $U_{i+1} = \text{qf}(U_{i+1})$; $V_{i+1} = \text{qf}(V_{i+1})$          // *Re-orthogonalize bases*
6:          $S_{i+1} = U_{i+1}^T X_{i+1} V_{i+1}$          // *Compute new S for next estimate of A*
7:          **if** condition **then**
8:                  // *Diagonalize S to obtain current estimate of singular vectors and values*
9:                  $[R_U, \Sigma_{i+1}, R_V] = \text{svd}(S_{i+1})$; $U_{i+1} = U_{i+1}R_U$; $V_{i+1} = V_{i+1}R_V$; $S_{i+1} = \Sigma_{i+1}$.
10:          **end if**
11: **end for**

---

matrix $A$ is fully observed, $U$ and $V$ can be randomly initialized, line 3 is not needed and in lines 4 and 6 we use $A$ instead of $X_{i+1}$ to update the subspaces. In this case, we have the classical two-sided subspace iteration for singular value decomposition. Lines 6-9 correspond to a Rayleigh-Ritz projection to obtain current approximations of singular vectors and singular values. It is known that if the initial columns of $U$ and $V$ are not orthogonal to any of the first $r$ left and right singular vectors of $A$ respectively, the algorithm converges to the dominant subspaces of $A$ [20, Theorem 5.1].

Back to the case when the matrix $A$ is not fully observed, the basic idea of Algorithm 1 is to use an approximation of $A$ in each iteration to update the subspaces $U$ and $V$ and then from the new $U$ and $V$, we can obtain a better approximation of $A$ for the next iteration. Line 3 is to compute a new estimate of $A$ by replacing all entries of $U_i S_i V_i^T$ at the known positions by the true values in $A$. The update in line 6 is to get the new $S_{i+1}$ based on recently computed subspaces. Diagonalizing

$S_{i+1}$ (lines 7-10) is optional for matrix completion. This step provides current approximations of the singular values which could be useful for several purposes such as in regularization or for convergence test. This comes with very little additional overhead, since $S_{i+1}$ is a small $r \times r$ matrix. Each iteration of Algorithm 1 can be seen as an approximation of an iteration of SVD-EM where a few matrix multiplications are used to update $U$ and $V$ instead of using a truncated SVD to compute the dominant subspaces of $X_{i+1}$. Recall that computing an SVD, e.g. by a Lanczos type procedure, requires several, possibly a large number of, matrix multiplications of this type.

We now discuss efficient implementations of Algorithm 1 and modifications to speed-up its convergence. First, the explicit computation of $X_{i+1}$ in line 3 is not needed. Let $\hat{X}_i = U_i S_i V_i^T$. Then $X_{i+1} = P_{\bar{\Omega}}(U_i S_i V_i^T) + A_\Omega = \hat{X}_i + E_i$, where $E_i = P_\Omega(A - \hat{X}_i)$ is a sparse matrix of errors at known entries which can be computed efficiently by exploiting the structure of $\hat{X}_i$. Assume that each $S_i$ is not singular (the non-singularity of $S_i$ will be discussed in Section 4). Then if we post-multiply the update of $U$ in line 4 by $S_i^{-1}$, the subspace remains the same, and the update becomes:

$$U_{i+1} = X_{i+1} V_i S_i^{-1} = (\hat{X}_i + E_i) V_i S_i^{-1} = U_i + E_i V_i S_i^{-1}, \tag{3}$$

The update of $V$ can also be efficiently implemented. Here, we make a slight change, namely $V_{i+1} = X_{i+1}^T U_i$ ($U_i$ instead of $U_{i+1}$). We observe that the convergence speed remains roughly the same (when $A$ is fully observed, the algorithm is a slower version of subspace iteration where the convergence rate is halved). With this change, we can derive an update to $V$ that is similar to (3),

$$V_{i+1} = V_i + E_i^T U_i S_i^{-T}, \tag{4}$$

We will point out in Section 3 that the updating terms $E_i V_i S_i^{-1}$ and $E_i^T U_i S_i^{-T}$ are related to the gradients of a matrix completion objective function on the Grassmann manifold. As a result, to improve the convergence speed, we can add an adaptive step size $t_i$ to the process, as follows:

$$U_{i+1} = U_i + t_i E_i V_i S_i^{-1} \quad \text{and} \quad V_{i+1} = V_i + t_i E_i^T U_i S_i^{-T}.$$

This is equivalent to using $\hat{X}_i + t_i E_i$ as the estimate of $A$ in each iteration. The step size can be computed using a heuristic adapted from [23]. Initially, $t$ is set to some initial value $t_0$ ($t_0 = 1$ in our experiments). If the error $\|E_i\|_F$ decreases compared to the previous step, $t$ is increased by a factor $\alpha$. Conversely, if the error increases, indicating that the step is too big, $t$ is reset to $t = t_0$.

The matrix $S_{i+1}$ can be computed efficiently by exploiting low-rank structures and the sparsity.

$$S_{i+1} = (U_{i+1}^T U_i) S_i (V_i^T V_{i+1}) + t_i U_{i+1}^T E_i V_{i+1} \tag{5}$$

There are also other ways to obtain $S_{i+1}$ once $U_{i+1}$ and $V_{i+1}$ are determined to improve the current approximation of $A$. For example we can solve the following quadratic program [14]:

$$S_{i+1} = \text{argmin}_S \|P_\Omega(A - U_{i+1} S V_{i+1}^T)\|_F^2 \tag{6}$$

We summarize the discussion in Algorithm 2. A sufficiently small error $\|E_i\|_F$ can be used as a

---
**Algorithm 2** GENERIC SUBSPACE ITERATION FOR INCOMPLETE MATRICES.
---
**Input:** Matrix $A_\Omega$, $\Omega$, and number $r$.
**Output:** Left and right dominant subspaces $U$ and $V$ and associated singular values.
  1: Initialize orthonormal matrices $U_0 \in \mathbb{R}^{m \times r}$ and $V_0 \in \mathbb{R}^{n \times r}$.
  2: **for** $i = 0,1,2,...$ **do**
  3:     Compute $E_i$ and appropriate step size $t_i$
  4:     $U_{i+1} = U_i + t_i E_i V_i S_i^{-1}$ and $V_{i+1} = V_i + t_i E_i^T U_i S_i^{-T}$
  5:     Orthonormalize $U_{i+1}$ and $V_{i+1}$
  6:     Find $S_{i+1}$ such that $P_\Omega(U_{i+1} S_{i+1} V_{i+1}^T)$ is close to $A_\Omega$ (e.g. via (5), (6)).
  7: **end for**
---

stoppping criterion. Algorithm 1 can be shown to be equivalent to LMaFit algorithm proposed in [23]. The authors in [23] also obtain results on local convergence of LMaFit. We will pursue a different approach here. The updates (3) and (4) are reminiscent of the gradient descent steps for minimizing matrix completion error on the Grassmann manifold that is introduced in [14] and the next section discusses the connection to optimization on the Grassmann manifold.

# 3 Optimization on the Grassmann manifold

In this section, we show that using a non-canonical Riemann metric on the Grassmann manifold, the gradient of the same objective function in [14] is of a form similar to (3) and (4). Based on this, improvements to the gradient descent algorithms can be made and exact recovery results similar to those of [14] can be maintained. The readers are referred to [1, 11] for details on optimization frameworks on matrix manifolds.

## 3.1 Gradients on the Grassmann manifold for matrix completion problem

Let $G(m, r)$ be the Grassmann manifold in which each point corresponds to a subspace of dimension $r$ in $\mathbb{R}^m$. One of the results of [14], is that under a few assumptions (to be addressed in Section 4), one can obtain with high probability the exact matrix $A$ by minimizing a regularized version of the function $F\colon G(m, r) \times G(n, r) \to \mathbb{R}$ defined below.

$$F(U, V) = \min_{S \in \mathbb{R}^{r \times r}} \mathcal{F}(U, S, V), \tag{7}$$

where $\mathcal{F}(U, S, V) = (1/2)\|P_\Omega(A - USV^T)\|_F^2$, $U \in \mathbb{R}^{m \times k}$ and $V \in \mathbb{R}^{n \times k}$ are orthonormal matrices. Here, we abuse the notation by denoting by $U$ and $V$ both orthonormal matrices as well as the points on the Grassmann manifold which they span. Note that $F$ only depends on the subspaces spanned by matrices $U$ and $V$. The function $F(U, V)$ can be easily evaluated by solving the quadratic minimization problem in the form of (6). If $G(m, r)$ is endowed with the canonical inner product $\langle W, W' \rangle = \mathrm{Tr}\,(W^T W')$, where $W$ and $W'$ are tangent vectors of $G(m, r)$ at $U$ (i.e. $W, W' \in \mathbb{R}^{m \times r}$ such that $W^T U = 0$ and $W'^T U = 0$) and similarly for $G(n, r)$, the gradients of $F(U, V)$ on the product manifold are:

$$\mathrm{grad}F_U(U, V) \;=\; (I - UU^T)P_\Omega(USV^T - A)VS^T \tag{8}$$
$$\mathrm{grad}F_V(U, V) \;=\; (I - VV^T)P_\Omega(USV^T - A)^T US. \tag{9}$$

In the above formulas, $(I - UU^T)$ and $(I - VV^T)$ are the projections of the derivatives $P_\Omega(USV^T - A)VS^T$ and $P_\Omega(USV^T - A)^T US$ onto the tangent space of the manifold at $(U, V)$. Notice that the derivative terms are very similar to the updates in (3) and (4). The difference is in the scaling factors where $\mathrm{grad}F_U$ and $\mathrm{grad}F_V$ use $S^T$ and $S$ while those in Algorithm 2 use $S^{-1}$ and $S^{-T}$.

Assume that $S$ is a diagonal matrix which can always be obtained by rotating $U$ and $V$ appropriately. $F(U, V)$ would change more rapidly when the columns of $U$ and $V$ corresponding to larger entries of $S$ are changed. The rate of change of $F$ would be approximately proportional to $S_{ii}^2$ when the $i$-th columns of $U$ and $V$ are changed, or in other words, $S^2$ gives us an approximate second order information of $F$ at the current point $(U, V)$. This suggests that the level set of $F$ should be similar to an "ellipse" with the shorter axes corresponding to the larger values of $S$. It is therefore compelling to use a scaled metric on the Grassmann manifold.

Consider the inner product $\langle W, W' \rangle_D = \mathrm{Tr}\,(DW^T W')$, where $D \in \mathbb{R}^{r \times r}$ is a symmetric positive definite matrix. We will derive the partial gradients of $F$ on the Grassmann manifold endowed with this scaled inner product. According to [11], $\mathrm{grad}F_U$ is the tangent vector of $G(m, r)$ at $U$ such that

$$\mathrm{Tr}\,(F_U^T W) = \langle (\mathrm{grad}F_U)^T, W \rangle_D, \tag{10}$$

for all tangent vectors $W$ at $U$, where $F_U$ is the partial derivative of $F$ with respect to $U$. Recall that the tangent vectors at $U$ are those $W$'s such that $W^T U = 0$. The solution of (10) with the constraints that $W^T U = 0$ and $(\mathrm{grad}F_U)^T U = 0$ gives us the gradient based on the scaled metric, which we will denote by $\mathrm{grad}_s F_U$ and $\mathrm{grad}_s F_V$.

$$\mathrm{grad}_s F_U(U, V) \;=\; (I - UU^T)F_U D^{-1} = (I - UU^T)P_\Omega(USV^T - A)VSD^{-1}. \tag{11}$$
$$\mathrm{grad}_s F_V(U, V) \;=\; (I - VV^T)F_V D^{-1} = (I - VV^T)P_\Omega(USV^T - A)^T USD^{-1}. \tag{12}$$

Notice the additional scaling $D$ appearing in these scaled gradients. Now if we use $D = S^2$ (still with the assumption that $S$ is diagonal) as suggested by the arguments above on the approximate shape of the level set of $F$, we will have $\mathrm{grad}_s F_U(U, V) = (I - UU^T)P_\Omega(USV^T - A)VS^{-1}$ and $\mathrm{grad}_s F_V(U, V) = (I - VV^T)P_\Omega(USV^T - A)^T USS^{-1}$ (note that $S$ depends on $U$ and $V$).

If $S$ is not diagonalized, we use $SS^T$ and $S^T S$ to derive $\text{grad}_s F_U$ and $\text{grad}_s F_V$ respectively, and the scalings appear exactly as in (3) and (4).

$$\text{grad}_s F_U(U,V) = (I - UU^T)P_\Omega(USV^T - A)VS^{-1} \tag{13}$$
$$\text{grad}_s F_V(U,V) = (I - VV^T)P_\Omega(USV^T - A)^T US^{-T} \tag{14}$$

This scaling can be interpreted as an adaptive preconditioning step similar to those that are popular in the scientific computing literature [4]. As will be shown in our experiments, this scaled gradient direction outperforms canonical gradient directions especially for ill-conditioned matrices.

The optimization framework on matrix manifolds allows to define several elements of the manifold in a flexible way. Here, we use the scaled-metric to obtain a good descent direction, while other operations on the manifold can be based on the canonical metric which has simple and efficient computational forms. The next two sections describe algorithms using scaled-gradients.

## 3.2 Gradient descent algorithms on the Grassmann manifold

Gradient descent algorithms on matrix manifolds are based on the update

$$U_{i+1} = R(U_i + t_i W_i) \tag{15}$$

where $W_i$ is the gradient-related search direction, $t_i$ is the step size and $R(U)$ is a retraction on the manifold which defines a projection of $U$ onto the manifold [1]. We use $R(U) = \text{span}(U)$ as the retraction on the Grassmann manifold where $\text{span}(U)$ is represented by $\text{qf}(U)$, which is the $Q$ factor in the QR factorization of $U$. Optimization on the product of two Grassmann manifolds can be done by treating each component as a coordinate component.

The step size $t$ can be computed in several ways, e.g., by a simple back-tracking method to find the point satisfying the Armijo condition [3]. Algorithm 3 is an outline of our gradient descent method for matrix completion. We let $\text{grad}_s F_U^{(i)} \equiv \text{grad}_s F_U(U_i, V_i)$ and $\text{grad}_s F_V^{(i)} \equiv \text{grad}_s F_V(U_i, V_i)$. In line 5, the exact $S_{i+1}$ which realizes $F(U_{i+1}, V_{i+1})$ can be computed according to (6). A direct method to solve (6) costs $O(|\Omega|r^4)$. Alternatively, $S_{i+1}$ can be computed approximately and we found that (5) is fast ($O((|\Omega| + m + n)r^2)$) and gives the same convergence speed. If (5) fails to yield good enough progress, we can always switch back to (6) and compute $S_{i+1}$ exactly. The subspace iteration and LMaFit can be seen as relaxed versions of this gradient descent procedure. The next section goes further and described the conjugate gradient iteration.

---

**Algorithm 3** GRADIENT DESCENT WITH SCALED-GRADIENT ON THE GRASSMANN MANIFOLD.

**Input:** Matrix $A_\Omega$, $\Omega$, and number $r$.
**Output:** $U$ and $V$ which minimize $F(U,V)$, and $S$ which realizes $F(U,V)$.
 1: Initialize orthonormal matrices $U_0$ and $V_0$.
 2: **for** $i = 0,1,2,...$ **do**
 3:     Compute $\text{grad}_s F_U^{(i)}$ and $\text{grad}_s F_V^{(i)}$ according to (13) and (14).
 4:     Find an appropriate step size $t_i$ and compute

$$(U_{i+1}, V_{i+1}) = (\text{qf}(U_i - t_i \text{grad}_s F_U^{(i)}), \text{qf}(V_i - t_i \text{grad}_s F_V^{(i)}))$$

 5:     Compute $S_{i+1}$ according to (6) (exact) or (5) (approximate).
 6: **end for**

---

## 3.3 Conjugate gradient method on the Grassmann manifold

In this section, we describe the conjugate gradient (CG) method on the Grassmann manifold based on the scaled gradients to solve the matrix completion problem. The main additional ingredient we need is vector transport which is used to transport the old search direction to the current point on the manifold. The transported search direction is then combined with the scaled gradient at the current point, e.g. by Polak-Ribiere formula (see [11]), to derive the new search direction. After this, a line search procedure is performed to find the appropriate step size along this search direction.

Vector transport can be defined using the Riemann connection, which in turn is defined based on the Riemann metric [1]. As mentioned at the end of Section 3.1, we will use the canonical metric to

derive vector transport when considering the natural quotient manifold structure of the Grassmann manifold. The tangent $W'$ at $U$ will be transported to $U + W$ as $T_{U+W}(W')$ where $T_U(W') = (I - UU^T)W'$. Algorithm 4 is a sketch of the resulting conjugate gradient procedure.

---

**Algorithm 4** CONJUGATE GRADIENT WITH SCALED-GRADIENT ON THE GRASSMANN MANIFOLD.

---

**Input:** Matrix $A_\Omega$, $\Omega$, and number $r$.
**Output:** $U$ and $V$ which minimize $F(U, V)$, and $S$ which realizes $F(U, V)$.
  1: Initialize orthonormal matrices $U_0$ and $V_0$.
  2: Compute $(\eta_0, \xi_0) = (\text{grad}_s F_U^{(0)}, \text{grad}_s F_V^{(0)})$.
  3: **for** $i = 0,1,2,...$ **do**
  4:     Compute a step size $t_i$ and compute $(U_{i+1}, V_{i+1}) = (\text{qf}(U_i + t_i\eta_i), \text{qf}(V_i + t_i\xi_i))$
  5:     Compute $\beta_{i+1}$ (Polak-Ribiere) and set

$$(\eta_{i+1}, \xi_{i+1}) = (-\text{grad}_s F_U^{(i)} + \beta_{i+1}T_{U_{i+1}}(\eta_i), -\text{grad}_s F_V^{(i)} + \beta_{i+1}T_{V_{i+1}}(\xi_i))$$

  6:     Compute $S_{i+1}$ according to (6) or (5).
  7: **end for**

---

## 4  Convergence and exact recovery of scaled-gradient descent methods

Let $A = U_*\Sigma_*V_*^T$ be the singular value decomposition of $A$, where $U_* \in \mathbb{R}^{m \times r}$, $V_* \in \mathbb{R}^{n \times r}$ and $\Sigma_* \in \mathbb{R}^{r \times r}$. Let us also denote $z = (U, V)$ a point on $G(m, r) \times G(n, r)$. Clearly, $z_* = (U_*, V_*)$ is a minimum of $F$. Assume that $A$ is incoherent [14]; $A$ has bounded entries and the minimum singular value of $A$ is bounded away from 0. Let $\kappa(A)$ be the condition number of $A$. It is shown that, if the number of observed entries is of order $O(\max\{\kappa(A)^2 n \log n, \kappa(A)^6 n\})$ then, with high probability, $F$ is well approximated by a parabola and $z_*$ is the unique stationary point of $F$ in a sufficiently small neighborhood of $z_*$ ([14, Lemma 6.4&6.5]). From these observations, given an initial point that is sufficiently close to $z_*$, a gradient descent procedure on $F$ (with an additional regularization term to keep the intermediate points incoherent) converges to $z_*$ and exact recovery is obtained. The singular value decomposition of a trimmed version of the observerd matrix $A_\Omega$ can give us the initial point that ensures convergence. The readers are referred to [14] for details.

From [14], let $G(U, V) = \sum_{i=1}^{m} G_1(\frac{\|U^{(i)}\|^2}{C_{inc}}) + \sum_{i=1}^{n} G_1(\frac{\|V^{(i)}\|^2}{C_{inc}})$, where $G_1(x) = 0$ if $x \leq 1$ and $G_1(x) = e^{(x-1)^2} - 1$ otherwise; $C_{inc}$ is a constant depending on the incoherence assumptions. We consider the regularized version of $F$: $\tilde{F}(U, V) = F(U, V) + \rho G(U, V)$, where $\rho$ is chosen appropriately so that $U$ and $V$ remain incoherent during the execution of the algorithm. We can see that $z_*$ is also the minimum of $\tilde{F}$. We will now show that the scaled-gradients of $\tilde{F}$ are well-defined during the iterations and they are indeed descent directions of $\tilde{F}$ and only vanish at $z_*$. As a result, the scaled-gradient-based methods can inherit all the convergence results in [14].

First, $S$ must be non-singular during the iterations for the scaled-gradients to be well-defined. As a corollary of Lemma 6.4 in [14], the extreme singular values of any intermediate $S$ are bounded by extreme singular values $\sigma_{min}^*$ and $\sigma_{max}^*$ of $\Sigma_*$: $\sigma_{max} \leq 2\sigma_{max}^*$ and $\sigma_{min} \geq \frac{1}{2}\sigma_{min}^*$. The second inequality implies that $S$ is well-conditioned during the iterations.

The scaled-gradient is the descent direction of $\tilde{F}$ as a direct result from the fact that it is indeed the gradient of $\tilde{F}$ based on a non-canonical metric. Moreover, by Lemma 6.5 in [14], $\|\text{grad}\tilde{F}(z)\|^2 \geq Cn\epsilon^2(\sigma_{min}^*)^4 d(z, z_*)^2$ for some constant $C$, where $\|.\|$ and $d(.,.)$ are the canonical norm and distance on the Grassmann manifold respectively. Based on this, a similar lower bound of $\|\text{grad}_s\tilde{F}\|$ can be derived. Let $D_1 = SS^T$ and $D_2 = S^TS$ be the scaling matrices. Then,

$$\begin{aligned}
\|\text{grad}_s\tilde{F}(z)\|^2 &= \|\text{grad}\tilde{F}_U(z)D_1^{-1}\|_F^2 + \|\text{grad}\tilde{F}_V(z)D_2^{-1}\|_F^2 \\
&\geq \sigma_{max}^{-2}(\|\text{grad}\tilde{F}_U(z)\|_F^2 + \|\text{grad}\tilde{F}_V(z)\|_F^2) \\
&\geq (2\sigma_{max}^*)^{-2}\|\text{grad}\tilde{F}(z)\|^2 \\
&\geq (2\sigma_{max}^*)^{-2}Cn\epsilon^2(\sigma_{min}^*)^4 d(z, z_*)^2 = C(\sigma_{min}^*)^4(2\sigma_{max}^*)^{-2}n\epsilon^2 d(z, z_*)^2.
\end{aligned}$$

Therefore, the scaled gradients only vanish at $z_*$ which means the scaled-gradient descent procedure must converge to $z_*$, which is the exact solution [3].

# 5 Experiments and results

The proposed algorithms were implemented in Matlab with some mex-routines to perform matrix multiplications with sparse masks. For synthesis data, we consider two cases: (1) *fully random low-rank matrices*, $A = \text{randn}(m, r) * \text{randn}(r, n)$ (in Matlab notations) whose singular values tend to be roughly the same; (2) *random low-rank matrices with chosen singular values* by letting $U = \text{qf}(\text{randn}(m, r))$, $V = \text{qf}(\text{randn}(n, r))$ and $A = USV^T$ where $S$ is a diagonal matrix with chosen singular values. The initializations of all methods are based on the SVD of $A_\Omega$.

First, we illustrate the improvement of scaled gradients over canonical gradients for steepest descent and conjugate gradient methods on $5000 \times 5000$ matrices with rank 5 (Figure 1). Note that Canon-Grass-Steep is OptSpace with our implementation. In this experiment, $S_i$ is obtained exactly using (6). The time needed for each iteration is roughly the same for all methods so we only present the results in terms of iteration counts. We can see that there are some small improvements for the fully random case (Figure 1a) since the singular values are roughly the same. The improvement is more substantial for matrices with larger condition numbers (Figure 1b).

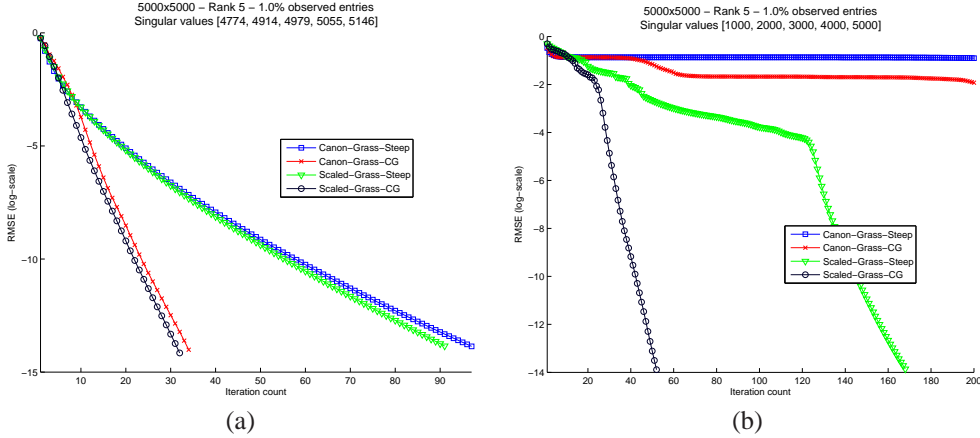

Figure 1: Log-RMSE for fully random matrix (a) and random matrix with chosen spectrum (b).

Now, we compare the relaxed version of the scaled conjugate gradient which uses (5) to compute $S_i$ (ScGrass-CG) to LMaFit [23], Riemann-CG [22], RTRMC2 [6] (trust region method with second order information), SVP [12] and GROUSE [5] (Figure 2). These methods are also implemented in Matlab with mex-routines similar to ours except for GROUSE which is entirely in Matlab (Indeed GROUSE does not use sparse matrix multiplication as other methods do). The subspace iteration method and the relaxed version of scaled steepest descent converge similarly to LMaFit, so we omit them in the graph. Note that each iteration of GROUSE in the graph corresponds to one pass over the matrix. It does not have exactly the same meaning as one iteration of other methods and is much slower with its current implementation. We use the best step sizes that we found for SVP and GROUSE. In terms of iteration counts, we can see that for the fully random case (upper row), RTRMC2 is the best while ScGrass-CG and Riemann-CG converge reasonably fast. However, each iteraton of RTRMC2 is slower so in terms of time, ScGrass-CG and Riemann-CG are the fastest in our experiments. When the condition number of the matrix is higher, ScGrass-CG converges fastest both in terms of iteration counts and execution time.

Finally, we test the algorithms on Jester-1 and MovieLens-100K datasets which are assumed to be low-rank matrices with noise (SVP and GROUSE are not tested because their step sizes need to be appropriately chosen). Similarly to previous work, for the Jester dataset we randomly select 4000 users and randomly withhold 2 ratings for each user for testing. For the MovieLens dataset, we use the common dataset prepared by [16], and keep 50% for training and 50% for testing. We run 100 different randomizations of Jester and 10 randomizations of MovieLens and average the results. We stop all methods early, when the change of RMSE is less than $10^{-4}$, to avoid overfitting. All methods stop well before one minute. The Normalized Mean Absolute Errors (NMAEs) [13] are reported in Table 1. ScGrass-CG is the relaxed scaled CG method and ScGrass-CG-Reg is the exact scaled CG method using a spectral-regularization version of $F$ proposed in

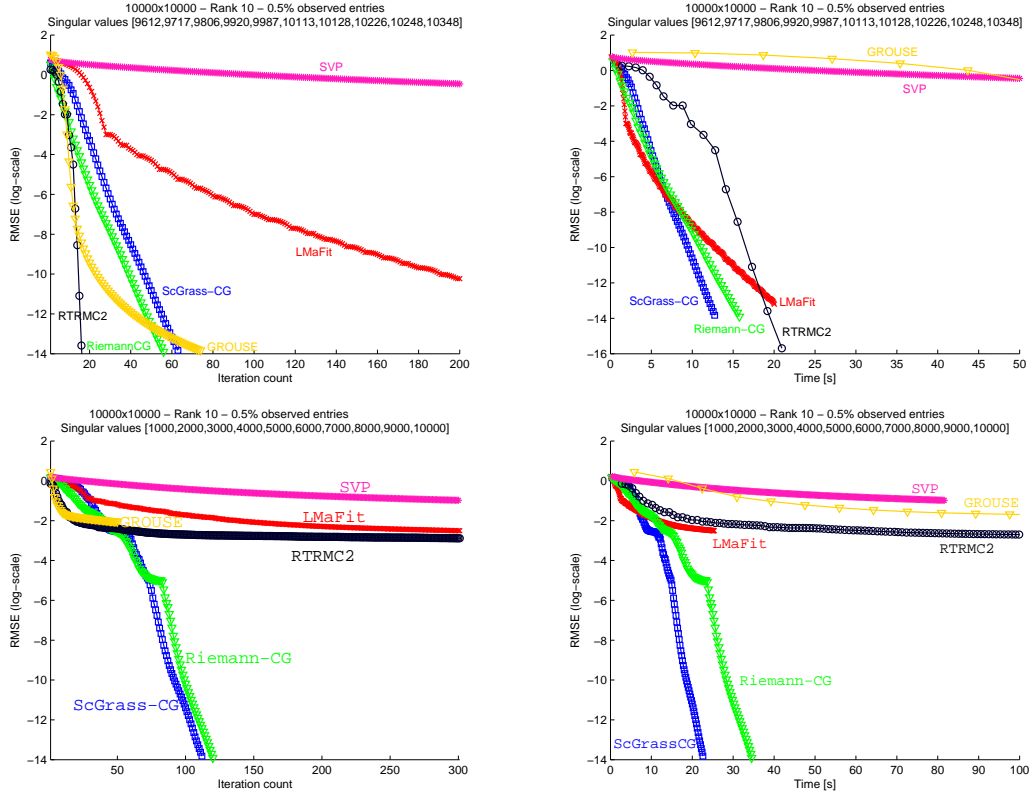

Figure 2: Log-RMSE. Upper row is fully random, lower row is random with chosen singular values.

| Rank | ScGrass-CG | ScGrass-CG-Reg | LMaFit | Riemann-CG | RTRMC2 |
|------|-----------|----------------|--------|------------|--------|
| 5 | 0.1588 | 0.1588 | 0.1588 | 0.1591 | 0.1588 |
| 7 | 0.1584 | 0.1584 | 0.1581 | 0.1584 | 0.1583 |
| 5 | 0.1808 | 0.1758 | 0.1828 | 0.1781 | 0.1884 |
| 7 | 0.1832 | 0.1787 | 0.1836 | 0.1817 | 0.2298 |

Table 1: NMAE on Jester dataset (first 2 rows) and MovieLens 100K. NMAEs for a random guesser are 0.33 on Jester and 0.37 on MovieLens 100K.

[13]: $\tilde{F}(U,V) = \min_S (1/2)(\|P_\Omega(USV^T - A)\| + \lambda\|S\|_F^2)$. All methods perform similarly and demonstrate overfitting when $k = 7$ for MovieLens. We observe that ScGrass-CG-Reg suffers the least from overfitting thanks to its regularization. This shows the importance of regularization for noisy matrices and motivates future work in this direction.

# 6   Conlusion and future work

The gradients obtained from a scaled metric on the Grassmann manifold can result in improved convergence of gradient methods on matrix manifolds for matrix completion while maintaining good global convergence and exact recovery guarantees. We have established a connection between scaled gradient methods and subspace iteration method for matrix completion. The relaxed versions of the proposed gradient methods, adapted from the subspace iteration, are faster than previously discussed algorithms, sometimes much faster depending on the conditionining of the data matrix. In the future, we will investigate if these relaxed versions achieve similar performance guarantees. We are also interested in exploring ways to regularize the relaxed versions to deal with noisy data. The convergence condition of OptSpace depends on $\kappa(A)^6$ and weakening this dependency for the proposed algorithms is also an interesting future direction.

**Acknowledgments**

This work was supported by NSF grants DMS-0810938 and DMR-0940218.

# References

[1] P.-A. Absil, R. Mahony, and R. Sepulchre. *Optimization Algorithms on Matrix Manifolds*. Princeton University Press, Princeton, NJ, 2008.

[2] Y. Amit, M. Fink, N. Srebro, and S. Ullman. Uncovering shared structures in multiclass classification. In *Proceedings of the 24th international conference on Machine learning*, ICML '07, pages 17–24, 2007.

[3] L. Armijo. Minimization of functions having Lipschitz continuous first partial derivatives. *Pacific Journal of Mathematics*, 16(1):1–3, 1966.

[4] J. Baglama, D. Calvetti, G. H. Golub, and L. Reichel. Adaptively preconditioned GMRES algorithms. *SIAM J. Sci. Comput.*, 20(1):243–269, December 1998.

[5] L. Balzano, R. Nowak, and B. Recht. Online identification and tracking of subspaces from highly incomplete information. In *Proceedings of Allerton*, September 2010.

[6] N. Boumal and P.-A. Absil. Rtrmc: A riemannian trust-region method for low-rank matrix completion. In *NIPS*, 2011.

[7] J-F. Cai, E. J. Candès, and Z. Shen. A singular value thresholding algorithm for matrix completion. *SIAM Journal on Optimization*, 20(4):1956–1982, 2010.

[8] E. Candes and T. Tao. The power of convex relaxation: Near-optimal matrix completion, 2009.

[9] P. Chen and D. Suter. Recovering the Missing Components in a Large Noisy Low-Rank Matrix: Application to SFM. *IEEE Transactions on Pattern Analysis and Machine Intelligence*, 26(8):1051–1063, 2004.

[10] W. Dai, E. Kerman, and O. Milenkovic. A geometric approach to low-rank matrix completion. *IEEE Transactions on Information Theory*, 58(1):237–247, 2012.

[11] A. Edelman, T. Arias, and S. T. Smith. The geometry of algorithms with orthogonality constraints. *SIAM J. Matrix Anal. Appl*, 20:303–353, 1998.

[12] P. Jain, R. Meka, and I. S. Dhillon. Guaranteed rank minimization via singular value projection. In *NIPS*, pages 937–945, 2010.

[13] R. Keshavan, A. Montanari, and S. Oh. Matrix completion from noisy entries. In Y. Bengio, D. Schuurmans, J. Lafferty, C. K. I. Williams, and A. Culotta, editors, *Advances in Neural Information Processing Systems 22*, pages 952–960. 2009.

[14] R. H. Keshavan, S. Oh, and A. Montanari. Matrix completion from a few entries. *CoRR*, abs/0901.3150, 2009.

[15] S. Ma, D. Goldfarb, and L. Chen. Fixed point and bregman iterative methods for matrix rank minimization. *Math. Program.*, 128(1-2):321–353, 2011.

[16] B. Marlin. Collaborative filtering: A machine learning perspective, 2004.

[17] R. Mazumder, T. Hastie, and R. Tibshirani. Spectral regularization algorithms for learning large incomplete matrices. *J. Mach. Learn. Res.*, 11:2287–2322, August 2010.

[18] B. Recht. A simpler approach to matrix completion. *CoRR*, abs/0910.0651, 2009.

[19] J. D. M. Rennie and N. Srebro. Fast maximum margin matrix factorization for collaborative prediction. In *In Proceedings of the 22nd International Conference on Machine Learning (ICML*, pages 713–719. ACM, 2005.

[20] Y. Saad. *Numerical Methods for Large Eigenvalue Problems- classics edition*. SIAM, Philadelpha, PA, 2011.

[21] N. Srebro and T. Jaakkola. Weighted low-rank approximations. In *In 20th International Conference on Machine Learning*, pages 720–727. AAAI Press, 2003.

[22] B. Vandereycken. Low-rank matrix completion by riemannian optimization. Technical report, Mathematics Section, Ecole Polytechnique Federale de de Lausanne, 2011.

[23] Z. Wen, W. Yin, and Y. Zhang. Solving a low-rank factorization model for matrix completion using a non-linear successive over-relaxation algorithm. In *CAAM Technical Report*. Rice University, 2010.

